# Potential-Based Agnostic Boosting

**Adam Tauman Kalai**
Microsoft Research
adum@microsoft.com

**Varun Kanade**
Harvard University
vkanade@fas.harvard.edu

## Abstract

We prove strong noise-tolerance properties of a potential-based boosting algorithm, similar to MadaBoost (Domingo and Watanabe, 2000) and SmoothBoost (Servedio, 2003). Our analysis is in the agnostic framework of Kearns, Schapire and Sellie (1994), giving polynomial-time guarantees in presence of arbitrary noise. A remarkable feature of our algorithm is that it can be implemented *without reweighting examples*, by randomly relabeling them instead. Our boosting theorem gives, as easy corollaries, alternative derivations of two recent nontrivial results in computational learning theory: agnostically learning decision trees (Gopalan *et al*, 2008) and agnostically learning halfspaces (Kalai *et al*, 2005). Experiments suggest that the algorithm performs similarly to MadaBoost.

## 1 Introduction

Boosting procedures attempt to improve the accuracy of general machine learning algorithms, through repeated executions on reweighted data. Aggressive reweighting of data may lead to poor performance in the presence of certain types of noise [1]. This has been addressed by a number of "robust" boosting algorithms, such as SmoothBoost [2, 3] and MadaBoost [4] as well as boosting by branching programs [5, 6]. Some of these algorithms are potential-based boosters, i.e., natural variants on AdaBoost [7], while others are perhaps less practical but have stronger theoretical guarantees in the presence of noise.

The present work gives a simple potential-based boosting algorithm with guarantees in the (arbitrary noise) agnostic learning setting [8, 9]. A unique feature of our algorithm, illustrated in Figure 1, is that it does not alter the distribution on unlabeled examples but rather it alters the labels. This enables us to prove a strong boosting theorem in which the weak learner need only succeed for one distribution on unlabeled examples. To the best of our knowledge, earlier weak-to-strong boosting theorems have always relied on the ability of the weak learner to succeed under arbitrary distributions. The utility of our boosting theorem is demonstrated by re-deriving two non-trivial results in computational learning theory, namely agnostically learning decision trees [10] and agnostically learning halfspaces [11], which were previously solved using very different techniques.

The main contributions of this paper are, first, giving the first provably noise-tolerant analysis of a potential-based boosting algorithm, and, second, giving a distribution-specific boosting theorem that does not require the weak learner to learn over all distributions on $x \in X$. This is in contrast to recent work by Long and Servedio, showing that convex potential boosters cannot work in the presence of random classification noise [12]. The present algorithm circumvents that impossibility result in two ways. First, the algorithm has the possibility of negating the current hypothesis and hence is not technically a standard potential-based boosting algorithm. Second, weak agnostic learning is more challenging than weak learning with random classification noise, in the sense that an algorithm which is a weak-learner in the random classification noise setting need not be a weak-learner in the agnostic setting.

**Related work.** There is a substantial literature on robust boosting algorithms, including algorithms already mentioned, MadaBoost, SmoothBoost, as well as LogitBoost [13], BrownBoost [14], Nad-

---

Simplified Boosting by Relabeling Procedure

Inputs: $(x_1, y_1), \ldots, (x_m, y_m) \in X \times \{-1, 1\}$, $T \geq 1$, and weak learner $W$.

Output: classifier $h : X \to \{-1, 1\}$.

1. Let $H^0 = \mathbf{0}$

2. For $t = 1, \ldots, T$ :

   (a) For $i = 1, \ldots, m$:
      - $w_i^t = \min\{1, \exp(-H^{t-1}(x_i)y_i)\}$
      - With probability $w_i^t$, set $\tilde{y}_i^t = y_i$, otherwise pick $\tilde{y}_i^t \in \{-1, 1\}$ randomly

   (b) $g^t = W\big((x_1, \hat{y}_1^t), \ldots, (x_m, \hat{y}_m^t)\big)$.

   (c) $h^t = \underset{g \in \{g^t, -\operatorname{sign}(H^{t-1})\}}{\operatorname{argmax}} \sum_i w_i^t y_i g(x_i)$. /* *possibly take negated hypothesis* */

   (d) $\gamma^t = \frac{1}{m} \sum_{i=1}^m w_i^t y_i h^t(x_i)$

   (e) $H^t(x) = H^{t-1}(x) + \gamma^t h^t(x)$

3. Output $h = \operatorname{sign}(H^T)$ as hypothesis.

---

Figure 1: Simplified Boosting by Relabeling Procedure. Each epoch, the algorithm runs the weak learner on *relabeled* data $\langle (x_i, \tilde{y}_i^t) \rangle_{i=1}^m$. In traditional boosting, on each epoch, $H^t$ is a linear combination of weak hypotheses. For our agnostic analysis, we also need to include the negated current hypothesis, $-\operatorname{sign}(H^{t-1}) : X \to \{-1, 1\}$, as a possible weak classifier. *In practice, to avoid adding noise, each example would be replaced with three weighted examples: $(x_i, y_i)$ with weight $w_i^t$, and $(x_i, \pm 1)$ each with weight $(1 - w_i^t)/2$.

aBoost [15] and others [16, 17], including extensive experimentation [18, 15, 19]. These are all simple boosting algorithms whose output is a weighted majority of classifiers. Many have been shown to have formal boosting properties (weak to strong PAC-learning) in a noiseless setting, or partial boosting properties in noisy settings. There has also been a line of work on boosting algorithms that provably boost from weak to strong learners either under agnostic or random classification noise, using branching programs [17, 20, 5, 21, 6]. Our results are stronger than those in the recent work of Kalai, Mansour, Verbin [6], for two main reasons. First, we propose a simple potential-based algorithm that can be implemented efficiently. Second, since we don't change the distribution over unlabeled examples, we can boost distribution-specific weak learners. In recent work, using a similar idea of relabeling, Kalai, Kanade and Mansour[22] proved that the class of DNFs is learnable in a one-sided error agnostic learning model. Their algorithm is essentially a simpler form of boosting.

**Experiments.** Our boosting procedure is quite similar to MadaBoost. The main differences are: (1) there is the possibility of using the negation of the current hypothesis at each step, (2) examples are relabeled rather than reweighted, and (3) the step size is slightly different. The goal of experiments was to understand how significant these differences may be in practice. Preliminary experimental results, presented in Section 5, suggest that all of these modifications are less important in practice than theory. Hence, the present simple analysis can be viewed as a theoretical justification for the noise-tolerance of MadaBoost and SmoothBoost.

## 1.1 Preliminaries

In the agnostic setting, we consider learning with respect to a distribution over $X \times Y$. For simplicity, we will take $X$ be to finite or countable and $Y = \{-1, 1\}$. Formally, learning is with respect to some class of functions, $\mathcal{C}$, where each $c \in \mathcal{C}$ is a binary classifier $c : X \to \{-1, 1\}$. There is an arbitrary distribution $\mu$ over $X$ and an arbitrary *target function* $f : X \to [-1, 1]$. Together these determine an arbitrary joint distribution $\mathcal{D} = \langle \mu, f \rangle$ over $X \times \{-1, 1\}$ where $\mathcal{D}(x, y) = \mu(x)\frac{1 + yf(x)}{2}$, i.e., $f(x) = \mathbb{E}_{\mathcal{D}}[y|x]$. The *error* and *correlation*[1] of a classifier $h : X \to \{-1, 1\}$ with respect to $\mathcal{D}$, are

respectively defined as,

$$\mathrm{err}(h, \mathcal{D}) = \Pr_{(x,y)\sim\mathcal{D}}[h(x) \neq y]$$

$$\mathrm{cor}(h, \mathcal{D}) = \mathbb{E}_{(x,y)\sim\mathcal{D}}[h(x)y] \; = \; \mathbb{E}_{x\sim\mu}[h(x)f(x)] \; = 1 - 2\,\mathrm{err}(h, \mathcal{D})$$

We will omit $\mathcal{D}$ when understood from context. The goal of the learning algorithm is to achieve error (equivalently correlation) arbitrarily close to that of the best classifier in $\mathcal{C}$, namely,

$$\mathrm{err}(\mathcal{C}) = \mathrm{err}(\mathcal{C}, \mathcal{D}) = \inf_{c\in\mathcal{C}}\mathrm{err}(c, \mathcal{D}); \qquad \mathrm{cor}(\mathcal{C}) = \mathrm{cor}(\mathcal{C}, \mathcal{D}) = \sup_{c\in\mathcal{C}}\mathrm{cor}(c, \mathcal{D})$$

A $\gamma$-weakly accurate classifier [23] for PAC (noiseless) learning is simply one whose correlation is at least $\gamma$ (for some $\gamma \in (0, 1)$). A different definition of weakly accurate classifier is appropriate in the agnostic setting. Namely, for some $\gamma \in (0, 1)$, $h : X \to \{-1, 1\}$ is said to be $\gamma$-*optimal for* $\mathcal{C}$ (and $\mathcal{D}$) if,

$$\mathrm{cor}(h, \mathcal{D}) \geq \gamma\,\mathrm{cor}(\mathcal{C}, \mathcal{D})$$

Hence, if the labels are totally random then a weak hypothesis need not have any correlation over random guessing. On the other hand, in a noiseless setting, where $\mathrm{cor}(\mathcal{C}) = 1$, this is equivalent to a $\gamma$-weakly accurate hypothesis. The goal is to boost from an algorithm capable of outputting $\gamma$-optimal hypotheses to one which outputs a nearly 1-optimal hypothesis, even for small $\gamma$.

Let $\mathcal{D}$ be a distribution over $X \times \{-1, 1\}$. Let $w : X \times \{-1, 1\} \to [0, 1]$ be a weighting function. We now define the distribution $\mathcal{D}$ *relabeled by* $w$, $R_{\mathcal{D},w}$. Procedurally, one can think of generating a sample from $R_{\mathcal{D},w}$ by drawing an example $(x, y)$ from $\mathcal{D}$, then with probability $w(x, y)$, outputting $(x, y)$ directly, and with probability $1 - w(x, y)$, outputting $(x, y')$ where $y'$ is uniformly random in $\{-1, 1\}$. Formally,

$$R_{\mathcal{D},w}(x, y) = \mathcal{D}(x, y)\left(w(x, y) + \frac{1 - w(x, y)}{2}\right) + \mathcal{D}(x, -y)\left(\frac{1 - w(x, -y)}{2}\right)$$

Note that $\mathcal{D}$ and $R_{\mathcal{D},w}$ have the same marginal distributions over unlabeled examples $x \in X$. Also, observe that, for any $\mathcal{D}$, $w$, and $h : X \to \mathbb{R}$,

$$\mathbb{E}_{(x,y)\sim R_{\mathcal{D},w}}[h(x)y] = \mathbb{E}_{(x,y)\sim\mathcal{D}}[h(x)yw(x, y)] \tag{1}$$

This can be seen by the procedural interpretation above. When $(x, y)$ is returned directly, which happens with probability $w(x, y)$, we get a contribution of $h(x)y$, but $\mathbb{E}[h(x)y'] = 0$ for uniform $y' \in \{-1, 1\}$.

It is possible to describe traditional supervised learning and active (query) learning in the same framework. A general $(m, q)$-learning algorithm is given $m$ unlabeled examples $\langle x_1, \ldots, x_m \rangle$, and may make $q$ label queries to a *query oracle* $L : X \to \{-1, 1\}$, and it outputs a classifier $h : X \to \{-1, 1\}$. The queries may be *active*, meaning that queries may only be made to training examples $x_i$, or *membership* queries meaning that arbitrary examples $x \in X$ may be queried. The active query setting where $q = m$ is the standard supervised learning setting where all $m$ labels may be queried. One can similarly model semi-supervised learning.

Since our boosting procedure does not change the distribution over unlabeled examples, it offers two advantages: (1) Agnostic weak learning may be defined with respect to a single distribution $\mu$ over unlabeled examples, and (2) The weak learning algorithms may be active (or use membership queries). In particular, the *agnostic weak learning hypothesis for* $\mathcal{C}$ *and* $\mu$ is that for any $f : X \to [-1, 1]$, given examples from $\mathcal{D} = \langle \mu, f \rangle$, the learner will output a $\gamma$-optimal classifier for $\mathcal{C}$. The advantages of this new definition are: (a) it is not with respect to every distribution on unlabeled examples (the algorithm may only have guarantees for certain distributions), and (b) it is more realistic as it does not assume noiseless data. Finding such a weak learner may be quite challenging since it has to succeed in the agnostic model (where no assumption is made on $f$), however it may be a bit easier in the sense that the learning algorithm need only handle one particular $\mu$.

**Definition 1.** *A learning algorithm is a* $(\gamma, \epsilon_0, \delta)$ *agnostic weak learner for* $\mathcal{C}$ *and* $\mu$ *over* $X$ *if, for any* $f : X \to [-1, 1]$*, with probability* $\geq 1 - \delta$ *over its random input, the algorithm outputs* $h : X \to [-1, 1]$ *such that, if* $\mathcal{D} = \langle \mu, f \rangle$*,*

$$\mathrm{cor}(h, \mathcal{D}) = \mathbb{E}_{x\sim\mu}[h(x)f(x)] \geq \gamma\left(\sup_{c\in\mathcal{C}}\mathbb{E}_{x\sim\mu}[c(x)f(x)]\right) - \epsilon_0$$

The $\epsilon_0$ parameter typically decreases quickly with the size of training data, e.g., $O(m^{-1/2})$. To see why it is necessary, consider a class $\mathcal{C} = \{c_1, c_2\}$ consisting of only two classifiers, and one of them has correlation $0$ and the other has minuscule positive correlation. Then, one cannot even identify which one has better correlation to within $O(m^{-1/2})$ using $m$ examples. Note that $\delta$ can easily made exponentially small (boosting confidence) using standard techniques.

Lastly, we define $\text{sign}(z)$ to be $1$ if $z \geq 0$ and $-1$ if $z < 0$.

## 2 Formal boosting procedure and main results

The formal boosting procedure we analyze is given in Figure 2.

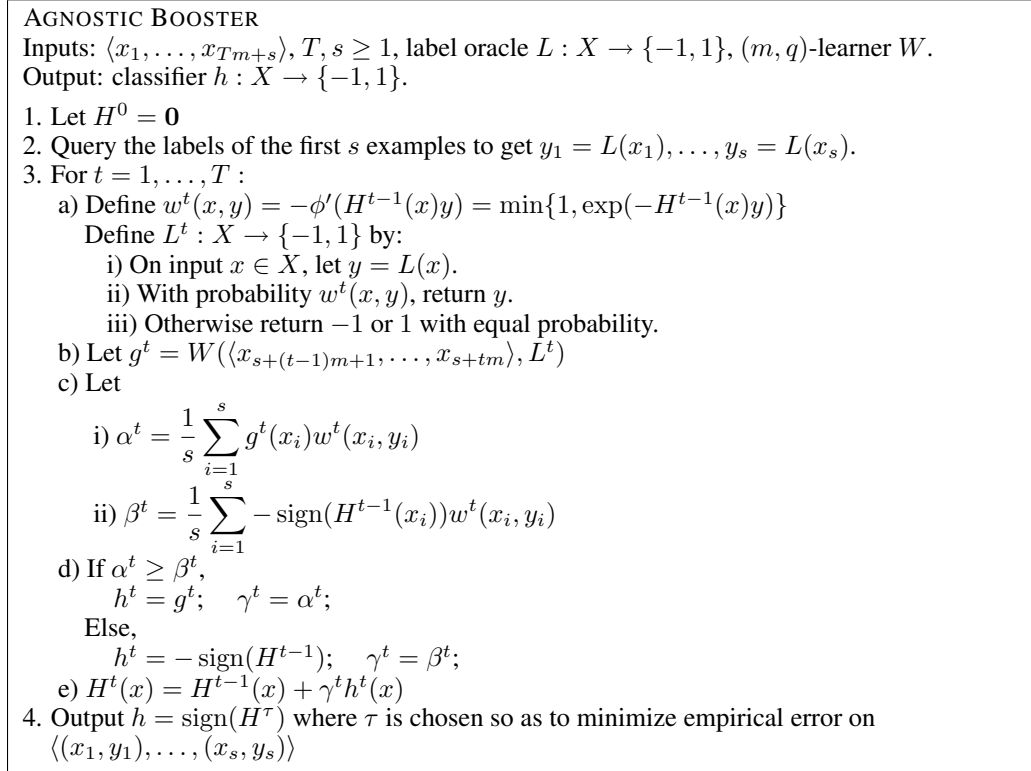

AGNOSTIC BOOSTER
Inputs: $\langle x_1, \ldots, x_{Tm+s} \rangle$, $T, s \geq 1$, label oracle $L : X \to \{-1, 1\}$, $(m, q)$-learner $W$.
Output: classifier $h : X \to \{-1, 1\}$.

1. Let $H^0 = \mathbf{0}$
2. Query the labels of the first $s$ examples to get $y_1 = L(x_1), \ldots, y_s = L(x_s)$.
3. For $t = 1, \ldots, T$:
  a) Define $w^t(x, y) = -\phi'(H^{t-1}(x)y) = \min\{1, \exp(-H^{t-1}(x)y)\}$
    Define $L^t : X \to \{-1, 1\}$ by:
      i) On input $x \in X$, let $y = L(x)$.
      ii) With probability $w^t(x, y)$, return $y$.
      iii) Otherwise return $-1$ or $1$ with equal probability.
  b) Let $g^t = W(\langle x_{s+(t-1)m+1}, \ldots, x_{s+tm} \rangle, L^t)$
  c) Let

  i) $\alpha^t = \dfrac{1}{s} \sum_{i=1}^{s} g^t(x_i) w^t(x_i, y_i)$

  ii) $\beta^t = \dfrac{1}{s} \sum_{i=1}^{s} -\text{sign}(H^{t-1}(x_i)) w^t(x_i, y_i)$
  d) If $\alpha^t \geq \beta^t$,
      $h^t = g^t; \quad \gamma^t = \alpha^t;$
    Else,
      $h^t = -\text{sign}(H^{t-1}); \quad \gamma^t = \beta^t;$
  e) $H^t(x) = H^{t-1}(x) + \gamma^t h^t(x)$
4. Output $h = \text{sign}(H^\tau)$ where $\tau$ is chosen so as to minimize empirical error on $\langle (x_1, y_1), \ldots, (x_s, y_s) \rangle$

Figure 2: Formal Boosting by Relabeling Procedure.

**Theorem 1.** *If $W$ is a $(\gamma, \epsilon_0, \delta)$ weak learner with respect to $\mathcal{C}$ and $\mu$, $s = \frac{200}{\gamma^2 \epsilon^2} \log\left(\frac{1}{\delta}\right)$, $T = \frac{29}{\gamma^2 \epsilon^2}$, Algorithm* AGNOSTIC BOOSTER *(Figure 2) with probability at least $1 - 4\delta T$ outputs a hypothesis $h$ satisfying:*

$$\text{cor}(h, \mathcal{D}) \geq \text{cor}(\mathcal{C}, \mathcal{D}) - \frac{\epsilon_0}{\gamma} - \epsilon$$

Recall that $\epsilon_0$ is intended to be very small, e.g., $O(m^{-1/2})$. Also note that the number of calls to the query oracle $L$ is $s$ plus $T$ times the number of calls made by the weak learner (if the weak learner is active, then so is the boosting algorithm). We show that two recent non-trivial results, viz. agnostically learning decision trees and agnostically learning halfspaces follow as corollaries to Theorem 1. The two results are stated below:

**Theorem 2** ([10]). *Let $\mathcal{C}$ be the class of binary decision trees on $\{-1, 1\}^n$ with at most $t$ leaves, and let $\mathcal{U}$ be the uniform distribution on $\{-1, 1\}^n$. There exists an algorithm that when given $t, n, \epsilon, \delta > 0$, and a label oracle for an arbitrary $f : \{-1, 1\}^n \to [-1, 1]$, makes $q = \text{poly}(nt/(\epsilon\delta))$ membership queries and, with probability $\geq 1 - \delta$, outputs $h : \{-1, 1\}^n \to \{-1, 1\}$ such that for $\mathcal{U}_f = \langle \mathcal{U}, f \rangle$, $\text{err}(h, \mathcal{U}_f) \leq \text{err}(\mathcal{C}, \mathcal{U}_f) + \epsilon$.*

**Theorem 3** ([11]). *For any fixed $\epsilon > 0$, there exists a univariate polynomial $p$ such that the following holds: Let $n \geq 1$, $\mathcal{C}$ be the class of halfspaces in $n$ dimensions, let $\mathcal{U}$ be the uniform distribution on $\{-1, 1\}^n$, and $f : \{-1, 1\}^n \rightarrow [-1, 1]$ be an arbitrary function. There exists a polynomial-time algorithm that, when given $m = p(n \log(1/\delta))$ labeled examples from $\mathcal{U}_f = \langle \mathcal{U}, f \rangle$, outputs a classifier $h : \{-1, 1\}^n \rightarrow \{-1, 1\}$ such that $\text{err}(h, \mathcal{U}_f) \leq \text{err}(\mathcal{C}, \mathcal{U}_f) + \epsilon$. (The algorithm makes no queries.)*

Note that a related theorem was shown for halfspaces over log-concave distributions over $X = \mathbb{R}^n$. The boosting approach here similarly generalizes to that case in a straightforward manner. This illustrates how, from the point of view of designing provably efficient agnostic learning algorithms, the current boosting procedure may be useful.

## 3  Analysis of Boosting Algorithm

This section is devoted to the analysis of algorithm AGNOSTIC BOOSTER (see Fig 2). As is standard, the boosting algorithm can be viewed as minimizing a convex potential function. However, the proof is significantly different than the analysis of AdaBoost [7], where they simply use the fact that the potential is an upper-bound on the error rate.

Our analysis has two parts. First, we define a *conservative* relabeling, such as the one we use, to be one which never relabels/downweights examples that the booster currently misclassifies. We show that for a conservative reweighting, either the weak learner will make progress, returning a hypothesis correlated with the relabeled distribution *or* $-\text{sign}(H^{t-1})$ will be correlated with the relabeled distribution.

Second, if we find a hypothesis correlated with the relabeled distribution, then the potential on round $t$ will be noticeably lower than that of round $t - 1$. This is essentially a simple gradient descent analysis, using a bound on the second derivative of the potential. Since the potential is between $0$ and $1$, it can only drop so many rounds. This implies that $\text{sign}(H^t)$ must be a near-optimal classifier for some $t$ (though the only sure way we have of knowing which one to pick is by testing accuracy on held-out data).

The potential function we consider, as in MadaBoost, is defined by $\phi : \mathbb{R} \rightarrow \mathbb{R}$,

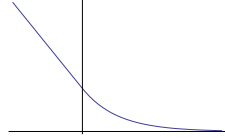

$$\phi(z) = \begin{cases} 1 - z & \text{if } z \leq 0 \\ e^{-z} & \text{if } z > 0 \end{cases}$$

Define the potential of a (real-valued) hypothesis $H$ with respect to a distribution $\mathcal{D}$ over $X \times \{-1, 1\}$ as:

$$\Phi(H, \mathcal{D}) = \mathop{\mathbb{E}}_{(x,y) \sim D} [\phi(yH(x))] \qquad (2)$$

Note that $\Phi(H^0, \mathcal{D}) = \Phi(\mathbf{0}, \mathcal{D}) = 1$. We will show that the potential decreases every round of the algorithm. Notice that the weights in the boosting algorithm correspond to the derivative of the potential, because $-\phi'(z) = \min\{1, \exp(-z)\} \in [0, 1]$. In other words, the weak learning step is essentially a gradient descent step.

We next state a key fact about agnostic learning in Lemma 1.

**Definition 2.** *Let $h : X \rightarrow \{-1, 1\}$ be a hypothesis. Then weighting function $w : X \times \{-1, 1\} \rightarrow [0, 1]$ is called* conservative *for $h$ if $w(x, -h(x)) = 1$ for all $x \in X$.*

Note that, if the hypothesis is $\text{sign}(H^t(x))$, then a weighting function defined by $-\phi'(H^t(x)y)$ is conservative if and only if $\phi'(z) = -1$ for all $z < 0$. We first show that relabeling according to a conservative weighting function is good in the sense that, if $h$ is far from optimal according to the original distribution, then after relabeling by $w$ it is even further from optimal.

**Lemma 1.** *For any distribution $\mathcal{D}$ over $X \times \{-1, 1\}$, classifiers $c, h : X \rightarrow \{-1, 1\}$, and any weighting function $w : X \times \{-1, 1\} \rightarrow [0, 1]$ conservative for $h$,*

$$\text{cor}(c, R_{\mathcal{D}, w}) - \text{cor}(h, R_{\mathcal{D}, w}) \geq \text{cor}(c, \mathcal{D}) - \text{cor}(h, \mathcal{D})$$

*Proof.* By the definition of correlation and eq. (1), $\mathrm{cor}(c, R_{\mathcal{D},w}) = \mathbb{E}_{\mathcal{D}}[c(x)yw(x,y)]$. Hence,

$$\mathrm{cor}(c, R_{\mathcal{D},w}) - \mathrm{cor}(h, R_{\mathcal{D},w}) = \mathrm{cor}(c, \mathcal{D}) - \mathrm{cor}(h, \mathcal{D}) - \mathop{\mathbb{E}}_{(x,y)\sim\mathcal{D}}[(c(x) - h(x))y(1 - w(x,y))]$$

Finally, consider two cases. In the first case, when $1 - w(x,y) > 0$, we have $h(x)y = 1$ while $c(x)y \leq 1$. The second case is $1 - w(x,y) = 0$. In either case, $(c(x) - h(x))y(1 - w(x,y)) \leq 0$. Thus the above equation implies the lemma. □

We will use Lemma 1 to show that the weak learner will return a useful hypothesis. The case in which the weak learner may not return a useful hypothesis is when $\mathrm{cor}(\mathcal{C}, R_{\mathcal{D},w}) = 0$, when the optimal classifier on the reweighted distribution has no correlation. This can happen, but in this case it means that either our current hypothesis is close to optimal, or $h = \mathrm{sign}(H^{t-1})$ is even worse than random guessing, and hence we can use its negation as a weak agnostic learner.

We next explain how a $\gamma$-optimal classifier on the reweighted distribution decreases the potential. We will use the following property linear approximation of $\phi$.

**Lemma 2.** *For any $x, \delta \in \mathbb{R}$, $|\phi(x + \delta) - \phi(x) - \phi'(x)\delta| \leq \delta^2/2$.*

*Proof.* This follows from Taylor's theorem and the fact the function $\phi$ is differentiable everywhere, and that the left and right second derivatives exist everywhere and are bounded by 1. □

Let $h^t : X \to \{-1, 1\}$ be the weak hypothesis that the algorithm finds on round $t$. This may either be the hypothesis returned by the weak learner $W$ or $-\mathrm{sign}(H^{t-1})$. The following lemma lower bounds the decrease in potential caused by adding $\gamma^t h^t$ to $H^{t-1}$. We will apply the following Lemma on each round of the algorithm to show that the potential decreases on each round, as long as the weak hypothesis $h^t$ has non-negligible correlation and $\gamma^t$ is suitably chosen.

**Lemma 3.** *Consider any function $H : X \to \mathbb{R}$, hypothesis $h : X \to [-1, 1]$, $\gamma \in \mathbb{R}$, and distribution $\mathcal{D}$ over $X \times \{-1, 1\}$. Let $\mathcal{D}' = R_{\mathcal{D},w}$ be the distribution $\mathcal{D}$ relabeled by $w(x,y) = -\phi'(yH(x))$. Then,*

$$\Phi(H, \mathcal{D}) - \Phi(H + \gamma h, \mathcal{D}) \geq \gamma \, \mathrm{cor}(h, \mathcal{D}') - \frac{\gamma^2}{2}$$

*Proof.* For any $(x,y) \in X \times \{-1, 1\}$, using Lemma 2 we know that:

$$\phi(H(x)y) - \phi((H(x) + \gamma h(x))y) \geq \gamma h(x)y(-\phi'(H(x)y)) - \frac{\gamma^2}{2}$$

In the step above we use the fact that $h(x)^2 y^2 \leq 1$. Taking expectation over $(x,y)$ from $\mathcal{D}$,

$$\Phi(H, \mathcal{D}) - \phi(H + \gamma h, \mathcal{D}) \geq \mathop{\mathbb{E}}_{(x,y)\sim\mathcal{D}}[h(x)y(-\phi'(H(x)y))] - \frac{\gamma^2}{2}$$

$$= \mathop{\mathbb{E}}_{(x,y)\sim\mathcal{D}'}[h(x)y] - \frac{\gamma^2}{2}$$

In the above we have used Eq. (1). We are done, by definition of $\mathrm{cor}(h, \mathcal{D}')$. □

Using all the above lemmas, we will show that the algorithm AGNOSTIC BOOSTER returns a hypothesis with correlation (or error) close to that of the best classifier from $\mathcal{C}$. We are now ready to prove the main theorem.

*Proof of Theorem 1.* Suppose $\exists c \in \mathcal{C}$ such that $\mathrm{cor}(c, \mathcal{D}) > \mathrm{cor}(\mathrm{sign}(H^{t-1}), \mathcal{D}) + \frac{\epsilon_0}{\gamma} + \epsilon$, then applying Lemma 1 to $H^{t-1}$ and setting $w^t(x,y) = -\phi'(H^{t-1}(x)y)$, we get that

$$\mathrm{cor}(c, R_{D,w^t}) > \mathrm{cor}(\mathrm{sign}(H^{t-1}), R_{\mathcal{D},w^t}) + \frac{\epsilon_0}{\gamma} + \epsilon \tag{3}$$

In this case we want to show that the algorithm successfully finds $h^t$ with $\mathrm{cor}(h^t, R_{\mathcal{D},w^t}) \geq \frac{\gamma\epsilon}{3}$.

Let $g^t$ be the hypothesis returned by the weak learner $W$. From Step 3c) in the algorithm:

$$\alpha^t = \frac{1}{s}\sum_{i=1}^{s} g(x_i)w^t(x_i, y_i); \quad \beta^t = \frac{1}{s}\sum_{i=1}^{s} -\operatorname{sign}(H^{t-1})(x_i)w^t(x_i, y_i)$$

When $s = \frac{200}{\gamma^2\epsilon^2}\log\left(\frac{1}{\delta}\right)$, by Chernoff-Hoeffding bounds we know that $\alpha^t$ and $\beta^t$ are within an additive $\frac{\gamma\epsilon}{20}$ of $\operatorname{cor}(g^t, R_{\mathcal{D},w^t})$ and $\operatorname{cor}(-\operatorname{sign}(H^{t-1}), R_{\mathcal{D},w^t})$ respectively with probability at least $1-2\delta$. As defined in Step 3d) in the algorithm, let $\gamma^t = \max(\alpha^t, \beta^t)$. We allow the algorithm to fail with probability $3\delta$ at this stage, possibly caused by the weak-learner and the estimation of $\alpha^t, \beta^t$.

Consider two cases: First that $\operatorname{cor}(c, R_{\mathcal{D},w^t}) \geq \frac{\epsilon_0}{\gamma} + \frac{\epsilon}{2}$, in this case by the weak learning assumption, $\operatorname{cor}(g^t, R_{\mathcal{D},w^t}) \geq \frac{\gamma\epsilon}{2}$. In the second case, if this does not hold, then $\operatorname{cor}(-\operatorname{sign}(H^{t-1}), R_{\mathcal{D},w^t}) \geq \frac{\epsilon}{2}$ using (3). Thus, even after taking into account the fact that the empirical estimates may be off from the true correlations by $\frac{\gamma\epsilon}{20}$, we get that $\operatorname{cor}(h^t, R_{\mathcal{D},w^t}) \geq \frac{\gamma\epsilon}{3}$ and that $|\gamma^t - \operatorname{cor}(h^t, R_{\mathcal{D},w^t})| \leq \frac{\gamma\epsilon}{20}$. Using this and Lemma 3, we get that by setting $H^t = H^{t-1} + \gamma^t h^t$ the potential decreases by at least $\frac{\gamma^2\epsilon^2}{29}$.

When $t = 0$ and $H^0 = \mathbf{0}$, $\Phi(H^0, \mathcal{D}) = 1$. Since for any $H : X \to \mathbb{R}$, $\Phi(H, \mathcal{D}) > 0$; we can have at most $T = \frac{29}{\gamma^2\epsilon^2}$ rounds. This guarantees that when the algorithm is run for $T$ rounds, on some round $t$ the hypothesis $\operatorname{sign}(H^t)$ will have correlation at least $\sup_{c\in\mathcal{C}}\operatorname{cor}(c, \mathcal{D}) - \frac{\epsilon_0}{\gamma} - \frac{2\epsilon}{3}$. For $s = \frac{200}{\gamma^2\epsilon^2}\log\left(\frac{1}{\delta}\right)$ the empirical estimate of the correlation of the constructed hypothesis on each round is within an additive $\frac{\epsilon}{6}$ of its true correlation, allowing a further failure probability of $\delta$ each round. Thus the final hypothesis $H^\tau$ which has the highest empirical correlation satisfies,

$$\operatorname{cor}(H^\tau, \mathcal{D}) \geq \sup_{c\in\mathcal{C}}\operatorname{cor}(c, \mathcal{D}) - \frac{\epsilon_0}{\gamma} - \epsilon$$

Since there is a failure probability of at most $4\delta$ on each round, the algorithm succeeds with probability at least $1 - 4T\delta$. $\qquad\square$

# 4 Applications

We show that recent agnostic learning analyses can be dramatically simplified using our boosting algorithm. Both of the agnostic algorithms are distribution-specific, meaning that they only work on one (or a family) of distributions $\mu$ over unlabeled examples.

## 4.1 Agnostically Learning Decision Trees

Recent work has shown how to agnostically learn polynomial-sized decision trees using membership queries, by an $L_1$ gradient-projection algorithm [10]. Here, we show that learning decision trees is quite simple using our distribution-specific boosting theorem and the Kushilevitz-Mansour membership query parity learning algorithm as a weak learner [24].

**Lemma 4.** *Running the KM algorithm, using $q = \operatorname{poly}(n, t, 1/\epsilon_0)$ queries, and outputting the parity with largest magnitude of estimated Fourier coefficient, is a $(\gamma = 1/t, \epsilon_0)$ agnostic weak learner for size-$t$ decision trees over the uniform distribution.*

The proof of this Lemma is simple using results in [24] and is given in Appendix A. Theorem 2 now follows easily from Lemma 4 and Theorem 1.

## 4.2 Agnostically Learning Halfspaces

In the case of learning halfspaces, the weak learner simply finds the degree-$d$ term, $\chi_S(x)$ with $|S| \leq d$, with greatest empirical correlation $\frac{1}{m}\sum_{i=1}^{m}\chi_S(x_i)y_i$ on a data set $(x_1, y_1), \ldots, (x_m, y_m)$. The following lemma is useful in analyzing it.

**Lemma 5.** *For any $\epsilon > 0$, there exists $d \geq 1$ such that the following holds. Let $n \geq 1$, $\mathcal{C}$ be the class of halfspaces in $n$ dimensions, let $\mathcal{U}$ be the uniform distribution on $\{-1,1\}^n$, and $f : \{-1,1\}^n \to [-1,1]$ be an arbitrary function. Then there exists a set $S \subseteq [n]$ of size $|S| \leq d = \frac{20}{\epsilon_0^4}$ such that $|\operatorname{cor}(\chi_S, \mathcal{U}_f)| \geq (\operatorname{cor}(\mathcal{C}, \mathcal{U}_f) - \epsilon_0)/n^d$.*

Using results from [25] the proofs of Lemma 5 and Theorem 3 are straightforward and are given in Appendix B.

## 5 Experiments

We performed preliminary experiments with the new boosting algorithm presented here on 8 datasets from UCI repository [26]. We converted multi-class problems into binary classification problems by arbitrarily grouping classes, and ran Adaboost, Madaboost and Agnostic Boost on these datasets, using stumps as weak learners. Since stumps can accept weighted examples, we passed the exact weighted distribution to the weak learner.

Our experiments were performed with *fractional relabeling*, which means the following. Rather than keeping the label with probability $w^t(x, y)$ and making it completely random with the remaining probability, we added both $(x, y)$ and $(x, -y)$ with weights $(1 + w^t(x, y))/2$ and $(1 - w^t(x, y))/2$ respectively. Experiments with random relabeling showed that random relabeling performs much worse than fractional relabeling.

Table 1 summarizes the final test error on the datasets. In the case of pima and german datasets, we observed overfitting and the reported test errors are the minimum test error observed for all the algorithms. In all other cases the test error rate at the end of round 500 is reported. Only pendigits had a test dataset, for the rest of the datasets we performed 10-fold cross validation. We also added random classification noise of 5%, 10% and 20% to the datasets and ran the boosting algorithms on the modified dataset.

| Dataset | No Added Noise | | | 5% noise | | | 10% Noise | | | 20% Noise | | |
|---|---|---|---|---|---|---|---|---|---|---|---|---|
| | Ada | Mada | Agn | Ada | Mada | Agn | Ada | Mada | Agn | Ada | Mada | Agn |
| sonar | 12.4 | 14.8 | 15.3 | 23.9 | 20.6 | 24.0 | 26.5 | 26.3 | 25.1 | 34.2 | 32.7 | 34.5 |
| ionosphere | 8.6 | 9.1 | 8.1 | 15.8 | 17.2 | 14.4 | 24.2 | 23.8 | 21.8 | 32 | 28.2 | 27.8 |
| pima | 23.7 | 23.0 | 23.6 | 26.1 | 24.9 | 25.7 | 27.6 | 26.4 | 26.7 | 34.3 | 34.5 | 34 |
| german | 23.1 | 23.6 | 23.1 | 28.5 | 27.7 | 27.5 | 29.0 | 29.5 | 30.0 | 35.0 | 34.5 | 35.1 |
| waveform | 10.4 | 10.2 | 10.3 | 14.9 | 15.0 | 13.9 | 20.1 | 19.2 | 19.1 | 27.9 | 27.3 | 27.1 |
| magic | 14.7 | 14.9 | 14.5 | 18.2 | 18.3 | 18.1 | 21.9 | 22.0 | 21.5 | 29.4 | 29.1 | 28.7 |
| letter | 17.4 | 18.2 | 18.3 | 20.9 | 21.4 | 21.5 | 24.6 | 24.9 | 25.2 | 31.4 | 31.8 | 31.6 |
| pendigits | 7.4 | 7.3 | 8.2 | 12.1 | 12.0 | 13.0 | 16.8 | 16.3 | 16.9 | 25.5 | 25.2 | 25.3 |

Table 1: Final test error rates of Adaboost, Madaboost and Agnostic Boosting on 8 datasets. The first column reports error rates on the original datasets, and the next three report errors on datasets with 5%, 10% and 20% classification noise added.

## 6 Conclusion

We show that potential-based agnostic boosting is possible in theory, and also that this may be done without changing the distribution over unlabeled examples. We show that non-trivial agnostic learning results, for learning decision trees and halfspaces, can be viewed as simple applications of our boosting theorem combined with well-known weak learners. Our analysis can be viewed as a theoretical justification of noise tolerance properties of algorithms like Madaboost and Smoothboost. Preliminary experiments show that the performance of our boosting algorithm is comparable to that of Madaboost and Adaboost. A more thorough empirical evaluation of our boosting procedure using different weak learners is part of future research.

## Footnotes

[1]This quantity is typically referred to as *edge* in the boosting literature. However, $\operatorname{cor}(h, \mathcal{D}) = 2 \operatorname{edge}(h, \mathcal{D})$ according to the standard notation, hence we use the notation cor.

## References

[1] T. G. Dietterich. An experimental comparison of three methods for constructing ensembles of decision trees: bagging, boosting, and randomization. *Machine Learning*, 40(2):139–158, 2000.

[2] R. Servedio. Smooth boosting and learning with malicious noise. *Journal of Machine Learning Research*, 4:633–648, 2003.

[3] D. Gavinsky. Optimally-smooth adaptive boosting and application to agnostic learning. *Journal of Machine Learning Research*, 4:101–117, 2003.

[4] C. Domingo and O. Watanabe. Madaboost: A modification of adaboost. In *Proceedings of the Thirteenth Annual Conference on Learning Theory*, pages 180–189, San Francisco, CA, USA, 2000. Morgan Kaufmann Publishers Inc.

[5] A. Kalai and R. Servedio. Boosting in the presence of noise. In *Proceedings of the 35th Annual Symposium on Theory of Computing (STOC)*, pages 196–205, 2003.

[6] A. T. Kalai, Y. Mansour, and E. Verbin. On agnostic boosting and parity learning. In *STOC '08: Proceedings of the 40th annual ACM symposium on Theory of computing*, pages 629–638, New York, NY, USA, 2008. ACM.

[7] Y. Freund and R. Schapire. Game theory, on-line prediction and boosting. In *Proceedings of the Ninth Annual Conference on Computational Learning Theory*, pages 325–332, 1996.

[8] D. Haussler. Decision theoretic generalizations of the pac model for neural net and other learning applications. *Inf. Comput.*, 100(1):78–150, 1992.

[9] M. Kearns, R. Schapire, and L. Sellie. Toward Efficient Agnostic Learning. *Machine Learning*, 17(2):115–141, 1994.

[10] P. Gopalan, A. T. Kalai, and A. R. Klivans. Agnostically learning decision trees. In *Proceedings of the 40th annual ACM symposium on Theory of computing*, pages 527–536, New York, NY, USA, 2008. ACM.

[11] A. T. Kalai, A. R. Klivans, Y. Mansour, and R. Servedio. Agnostically learning halfspaces. In *Proc. $46^{th}$ IEEE Symp. on Foundations of Computer Science (FOCS'05)*, 2005.

[12] P. M. Long and R. A. Servedio. Random classification noise defeats all convex potential boosters. In *ICML*, pages 608–615, 2008.

[13] J. Friedman, T. Hastie, and R. Tibshirani. Additive logistic regression: a statistical view of boosting. *Annals of Statistics*, 28:2000, 1998.

[14] Y. Freund. An adaptive version of the boost-by-majority algorithm. In *Proceedings of the Twelfth Annual Conference on Computational Learning Theory*, pages 102–113, 1999.

[15] M. Nakamura, H. Nomiya, and K. Uehara. Improvement of boosting algorithm by modifying the weighting rule. *Annals of Mathematics and Artificial Intelligence*, 41(1):95–109, 2004.

[16] T. Bylander and L. Tate. Using validation sets to avoid overfitting in adaboost. In G. Sutcliffe and R. Goebel, editors, *FLAIRS Conference*, pages 544–549. AAAI Press, 2006.

[17] S. Ben-David, P. M. Long, and Y. Mansour. Agnostic boosting. In *Proceedings of the 14th Annual Conference on Computational Learning Theory, COLT 2001*, volume 2111 of *Lecture Notes in Artificial Intelligence*, pages 507–516. Springer, 2001.

[18] R. A. McDonald, D. J. Hand, and I. A. Eckley. An empirical comparison of three boosting algorithms on real data sets with artificial class noise. In T. Windeatt and F. Roli, editors, *Multiple Classifier Systems*, volume 2709 of *Lecture Notes in Computer Science*, pages 35–44. Springer, 2003.

[19] J. K. Bradley and R. Schapire. Filterboost: Regression and classification on large datasets. In J.C. Platt, D. Koller, Y. Singer, and S. Roweis, editors, *Advances in Neural Information Processing Systems 20*, pages 185–192. MIT Press, Cambridge, MA, 2008.

[20] Y. Mansour and D. McAllester. Boosting using branching programs. *Journal of Computer and System Sciences*, 64(1):103–112, 2002.

[21] P. M. Long and R. A. Servedio. Adaptive martingale boosting. In *NIPS*, pages 977–984, 2008.

[22] A. T. Kalai, V. Kanade, and Y. Mansour. Reliable agnostic learning. In *COLT '09: Proceedings of the 22nd annual conference on learning theory*, 2009.

[23] M. Kearns and L. Valiant. Cryptographic limitations on learning Boolean formulae and finite automata. *Journal of the ACM*, 41(1):67–95, 1994.

[24] E. Kushilevitz and Y. Mansour. Learning decision trees using the Fourier spectrum. *SIAM J. on Computing*, 22(6):1331–1348, 1993.

[25] A. Klivans, R. O'Donnell, and R. Servedio. Learning intersections and thresholds of halfspaces. *Journal of Computer & System Sciences*, 68(4):808–840, 2004.

[26] A. Asuncion and D. J. Newman. UCI Machine Learning Repository [http://www.ics.uci.edu/~mlearn/MLRepository.html] Irvine, CA: University of California, School of Information and Computer Science, 2007.

